# A Rotation and Translation Invariant Discrete Saliency Network

**Lance R. Williams**
Dept. of Computer Science
Univ. of New Mexico
Albuquerque, NM 87131

**John W. Zweck**
Dept. of CS and EE
Univ. of Maryland Baltimore County
Baltimore, MD 21250

## Abstract

We describe a neural network which enhances and completes salient closed contours. Our work is different from all previous work in three important ways. First, like the input provided to V1 by LGN, the input to our computation is isotropic. That is, the input is composed of spots not edges. Second, our network computes a well defined function of the input based on a distribution of closed contours characterized by a random process. Third, even though our computation is implemented in a discrete network, its output is invariant to continuous rotations and translations of the input pattern.

## 1 Introduction

There is a long history of research on neural networks inspired by the structure of visual cortex whose functions have been described as contour completion, saliency enhancement, orientation sharpening, or segmentation[6, 7, 8, 9, 12]. A similiar network has been proposed as a model of visual hallucinations[1]. In this paper, we describe a neural network which enhances and completes salient closed contours. Our work is different from all previous work in three important ways. First, like the input provided to V1 by LGN, the input to our computation is isotropic. That is, the input is composed of spots not edges. Second, our network computes a well defined function of the input based on a distribution of closed contours characterized by a random process. Third, even though our computation is implemented in a discrete network, its output is invariant to continuous rotations and translations of the input pattern.

There are two important properties which a computation must possess if it is to be invariant to rotations and translations, i.e., Euclidean invariant. First, the input, the output, and all intermediate representations must be Euclidean invariant. Second, all transformations of these representations must also be Euclidean invariant. The models described in [6, 7, 8, 9, 12] are not Euclidean invariant, first and foremost, because their input representations are not Euclidean invariant. That is, not all rotations and translations of the input can be represented equally well. This problem is often skirted by researchers by choosing input patterns which match particular choices of sampling rate and phase. For example, Li [7] used only six samples in orientation (including $0°$) and Heitger and von der Heydt[5] only twelve (including $0°$, $60°$ and $120°$). Li's first test pattern was a dashed line of orientation, $0°$, while Heitger and von der Heydt used a Kanizsa Triangle with sides of $0°$, $60°$, and

120° orientation. There is no reason to believe that the experimental results they showed would be similiar if the input patterns were rotated by as little as 5°. To our knowledge, no researcher in this area has ever commented on this problem before.

## 2  A continuum formulation of the saliency problem

The following section reviews the continuum formulation of the contour completion and saliency problem as described in Williams and Thornber[11].

### 2.1  Shape distribution

Mumford[3] observed that the probability distribution of object boundary shapes could be modeled by a *Fokker-Planck* equation of the following form:

$$\frac{\partial p}{\partial t} = -\cos\theta \frac{\partial p}{\partial x} - \sin\theta \frac{\partial p}{\partial y} + \frac{\sigma^2}{2}\frac{\partial^2 p}{\partial \theta^2} - \frac{1}{\tau}p. \tag{1}$$

where $p(\vec{x}, \theta\,;\,t)$ is the probability that a particle is located at position, $\vec{x} = (x, y)$, and is moving in direction, $\theta$, at time, $t$. This partial differential equation can be viewed as a set of independent *advection* equations in $x$ and $y$ (the first and second terms) coupled in the $\theta$ dimension by the *diffusion* equation (the third term). The advection equations translate probability mass in direction, $\theta$, with unit speed, while the diffusion term models the Brownian motion in direction, with *diffusion parameter*, $\sigma$. The combined effect of these three terms is that particles tend to travel in straight lines, but over time they drift to the left or right by an amount proportional to $\sigma^2$. Finally, the effect of the fourth term is that particles decay over time, with a half-life given by the decay constant, $\tau$.

### 2.2  The propagators

The Green's function, $G(\vec{x}, \theta\,;\,t_1 \mid \vec{u}, \phi\,;\,t_0)$, gives the probability that a particle observed at position, $\vec{u}$, and direction, $\phi$, at time, $t_0$, will later be observed at position, $\vec{x}$, and direction, $\theta$, at time, $t_1$. It is the solution, $p(\vec{x}, \theta\,;\,t_1)$, of the Fokker-Planck initial value problem with initial value, $p(\vec{x}, \theta\,;\,t_0) = \delta(\vec{x} - \vec{u})\delta(\theta - \phi)$ where $\delta$ is the Dirac delta function. The Green's function is used to define two *propagators*. The long-time propagator:

$$P_0(\vec{x}, \theta \mid \vec{u}, \phi) = \int_0^\infty dt\, \chi(t)\, G(\vec{x}, \theta\,;\,t \mid \vec{u}, \phi\,;\,0) \tag{2}$$

gives the probability that $(\vec{x}, \theta)$ and $(\vec{u}, \phi)$ are distinct edges from the boundary of a single object. [1] The short-time propagator:

$$P_1(\vec{x}, \theta \mid \vec{u}, \phi) = \int_0^\infty dt\, [1 - \chi(t)]\, G(\vec{x}, \theta\,;\,t \mid \vec{u}, \phi\,;\,0) \tag{3}$$

gives the probability that $(\vec{x}, \theta)$ and $(\vec{u}, \phi)$ are from the boundary of a single object but are really the same edge. In both of these propagators, $\chi(.)$ is a cut-off function with $\chi(0) = 0$ and $\lim_{t \to \infty} \chi(t) = 1$:

$$\chi(t) = \frac{1}{2}\left[1 + \frac{2}{\pi}\operatorname{atan}\left(\mu\left[\frac{t}{\Delta} - \alpha\right]\right)\right]. \tag{4}$$

The cut-off function is characterized by three parameters, $\alpha$, $\mu$, and $\Delta$. The parameter, $\alpha$, specifies where the cut-off is and $\mu$ specifies how hard hard it is. The parameter, $\Delta$, is the scale of the edge detection process.

## 2.3 Eigenfunctions

The integral linear operator, $Q(.)$, combines three sources of information: 1) the probability that two edges belong to the same object; 2) the probability that the two edges are distinct; and 3) the probability that the two edges exist. It is defined as follows:

$$Q(\vec{x}, \theta \mid \vec{u}, \phi) = b(\vec{x})^{\frac{1}{2}} P_0(\vec{x}, \theta \mid \vec{u}, \phi) b(\vec{u})^{\frac{1}{2}} \tag{5}$$

where the *input bias function*, $b(\vec{x})$, gives the probability that an edge exists at $\vec{x}$. As described in Williams and Thornber[11], the right and left eigenfunctions, $s(.)$ and $\bar{s}(.)$, of $Q(.)$ with largest positive real eigenvalue, $\lambda$, play a central role in the computation of saliency:

$$\lambda s(\vec{x}, \theta) = \int \int \int_{\mathbf{R}^2 \times S^1} d\vec{u} d\phi \ Q(\vec{x}, \theta \mid \vec{u}, \phi) s(\vec{u}, \phi) \tag{6}$$

$$\lambda \bar{s}(\vec{x}, \theta) = \int \int \int_{\mathbf{R}^2 \times S^1} d\vec{u} d\phi \ \bar{s}(\vec{u}, \phi) Q(\vec{u}, \phi \mid \vec{x}, \theta). \tag{7}$$

Because $Q(.)$ is invariant under a transformation which reverses the order and direction of its arguments:

$$Q(\vec{x}, \theta \mid \vec{u}, \phi) = Q(\vec{u}, \phi + \pi \mid \vec{x}, \theta + \pi) \tag{8}$$

the right and left eigenfunctions are related as follows:

$$\bar{s}(\vec{x}, \theta) = s(\vec{x}, \theta + \pi). \tag{9}$$

## 2.4 Stochastic completion field

The magnitude of the *stochastic completion field*, $c(\vec{u}, \phi)$, equals the probability that a closed contour satisfying a subset of the constraints exists at $(\vec{u}, \phi)$. It is the sum of three terms:

$$c(\vec{u}, \phi) = \frac{p_0(\vec{u}, \phi) \bar{p}_0(\vec{u}, \phi) + p_0(\vec{u}, \phi) \bar{p}_1(\vec{u}, \phi) + p_1(\vec{u}, \phi) \bar{p}_0(\vec{u}, \phi)}{\lambda \int \int \int_{\mathbf{R}^2 \times S^1} d\vec{x} d\theta \ s(\vec{x}, \theta) \bar{s}(\vec{x}, \theta)} \tag{10}$$

where $p_m(\vec{u}, \phi)$ is a *source field*, and $\bar{p}_m(\vec{u}, \phi)$ is a *sink field*:

$$p_m(\vec{u}, \phi) = \int \int \int_{\mathbf{R}^2 \times S^1} d\vec{x} d\theta \ P_m(\vec{u}, \phi \mid \vec{x}, \theta) b(\vec{x})^{\frac{1}{2}} s(\vec{x}, \theta) \tag{11}$$

$$\bar{p}_m(\vec{u}, \phi) = \int \int \int_{\mathbf{R}^2 \times S^1} d\vec{x} d\theta \ \bar{s}(\vec{x}, \theta) b(\vec{x})^{\frac{1}{2}} P_m(\vec{x}, \theta \mid \vec{u}, \phi). \tag{12}$$

The purpose of writing $c(\vec{u}, \phi)$ in this way is to remove the contribution, $p_1(\vec{u}, \phi) \bar{p}_1(\vec{u}, \phi)$, of closed contours at scales smaller than $\Delta$ which would otherwise dominate the completion field. Given the above expression for the completion field, it is clear that the key problem is computing the eigenfunction, $s(.)$, of $Q(.)$ with largest positive real eigenvalue. To accomplish this, we can use the well known power method (see [4]). In this case, the power method involves repeated application of the linear operator, $Q(.)$, to the function, $s(.)$, followed by normalization:

$$s^{m+1}(\vec{x}, \theta) = \frac{\int \int \int_{\mathbf{R}^2 \times S^1} d\vec{u} d\phi \ Q(\vec{x}, \theta \mid \vec{u}, \phi) s^m(\vec{u}, \phi)}{\int \int \int_{\mathbf{R}^2 \times S^1} \int \int \int_{\mathbf{R}^2 \times S^1} d\vec{x} d\theta d\vec{u} d\phi \ Q(\vec{x}, \theta \mid \vec{u}, \phi) s^m(\vec{u}, \phi)}. \tag{13}$$

In the limit, as $m$ gets very large, $s^{(m+1)}(\vec{x}, \theta)$ converges to the eigenfunction of $Q(.)$, with largest positive real eigenvalue. We observe that the above computation can be considered a continuous state, discrete time, recurrent neural network.

## 3  A discrete implementation of the continuum formulation

The continuous functions comprising the state of the computation are represented as weighted sums of a finite set of *shiftable-twistable* basis functions. The weights form the coefficient vectors for the functions. The computation we describe is biologically plausible in the sense that all transformations of state are effected by linear transformations (or other vector parallel operations) on the coefficient vectors.

## 3.1 Shiftable-twistable bases

The input and output of the above computation are functions defined on the continuous space, $\mathbf{R}^2 \times S^1$, of positions in the plane, $\mathbf{R}^2$, and directions in the circle, $S^1$. For such computations, the important symmetry is determined by those transformations, $T_{\vec{x}_0, \theta_0}$, of $\mathbf{R}^2 \times S^1$, which perform a shift in $\mathbf{R}^2$ by $\vec{x}_0$, followed by a twist in $\mathbf{R}^2 \times S^1$ through an angle, $\theta_0$. A *twist* through an angle, $\theta_0$, consists of two parts: (1) a rotation, $R_{\theta_0}$, of $\mathbf{R}^2$ and (2) a translation in $S^1$, both by $\theta_0$. The symmetry, $T_{\vec{x}_0, \theta_0}$, which is called a *shift-twist transformation*, is given by the formula,

$$T_{(\vec{x}_0, \theta_0)}(\vec{x}, \theta) \;=\; (R_{\theta_0}(\vec{x} - \vec{x}_0), \, \theta - \theta_0). \tag{14}$$

A visual computation, $C$, on $\mathbf{R}^2 \times S^1$ is called *shift-twist invariant* if, for all $(\vec{x}_0, \theta_0) \in \mathbf{R}^2 \times S^1$, a shift-twist of the input by $(\vec{x}_0, \theta_0)$ produces an identical shift-twist of the output. This property can be depicted in the following commutative diagram:

$$
\begin{array}{ccc}
b(\vec{x}, \theta) & \overset{C}{\to} & c(\vec{x}, \theta) \\
\downarrow T_{\vec{x}_0, \theta_0} & & \downarrow T_{\vec{x}_0, \theta_0} \\
b(R_{\theta_0}(\vec{x} - \vec{x}_0), \theta - \theta_0) & \overset{C}{\to} & c(R_{\theta_0}(\vec{x} - \vec{x}_0), \theta - \theta_0)
\end{array}
$$

where $b(.)$ is the input, $c(.)$, is the output, $\overset{C}{\to}$ is the computation, and $\overset{T_{\vec{x}_0, \theta_0}}{\to}$ is the shift-twist transformation. Correspondingly, we define a *shiftable-twistable basis*[2] of functions on $\mathbf{R}^2 \times S^1$ to be a set of functions on $\mathbf{R}^2 \times S^1$ with the property that whenever a function, $f(\vec{x}, \theta)$, is in their span, then so is $f(T_{\vec{x}_0, \theta_0}(\vec{x}, \theta))$, for every choice of $(\vec{x}_0, \theta_0)$ in $\mathbf{R}^2 \times S^1$. As such, the notion of a shiftable-twistable basis on $\mathbf{R}^2 \times S^1$ generalizes that of a shiftable-steerable basis on $\mathbf{R}^2$[2, 10].

Shiftable-twistable bases can be constructed as follows. Let $\Psi(\vec{x}, \theta)$ be a function on $\mathbf{R}^2 \times S^1$ which is periodic (with period $X$) in both spatial variables, $\vec{x}$. In analogy with the definition of a shiftable-steerable function on $\mathbf{R}^2$, we say that $\Psi$ is *shiftable-twistable* on $\mathbf{R}^2 \times S^1$ if there are integers, $K$ and $M$, and interpolation functions, $a_{\vec{k}, m}(\vec{x}_0, \theta_0)$, such that for each $(\vec{x}_0, \theta_0) \in \mathbf{R}^2 \times S^1$, the shift-twist of $\Psi$ by $(\vec{x}_0, \theta_0)$ is a linear combination of a finite number of basic shift-twists of $\Psi$ by amounts $(\vec{k}\Delta, m\Delta_\theta)$, i.e., if

$$\Psi(T_{\vec{x}_0, \theta_0}(\vec{x}, \theta)) = \sum_{\vec{k}, m} a_{\vec{k}, m}(\vec{x}_0, \theta_0) \Psi(T_{\vec{k}\Delta, m\Delta_\theta}(\vec{x}, \theta)). \tag{15}$$

Here $\Delta = X/K$ is the *basic shift amount* and $\Delta_\theta = 2\pi/M$ is the *basic twist amount*. The sum in equation (15) is taken over all pairs of integers, $\vec{k} = (k_x, k_y)$, in the range, $0 \le k_x, k_y < K$, and all integers, $m$, in the range, $0 \le m < M$.

The Gaussian-Fourier basis is the product of a shiftable-steerable basis of Gaussians in $\vec{x}$ and a Fourier series basis in $\theta$. For the experiments in this paper, the standard deviation of the Gaussian basis function, $g(\vec{x}) = \frac{1}{\Delta} e^{-\|\vec{x}\|^2/2\Delta^2}$, equals the basic shift amount, $\Delta$. We regard $g(\vec{x})$ as a periodic function of period, $X$, which is chosen to be much larger than $\Delta$, so that $g(X/2, X/2)$ and its derivatives are essentially zero. For each frequency, $\omega$, and shift amount, $\Delta$ (where $K = X/\Delta$ is an integer), we define the *Gaussian-Fourier basis functions*, $\Psi_{\vec{k}, \omega}$, by

$$\Psi_{\vec{k}, \omega}(\vec{x}, \theta) \;=\; g(\vec{x} - \vec{k}\Delta) \, e^{i\omega\theta}. \tag{16}$$

Zweck and Williams[13] showed that the Gaussian-Fourier basis is shiftable-twistable.

## 3.2 Power method update formula

Suppose that $s^{(m)}(\vec{x}, \theta)$ can be represented in the Gaussian-Fourier basis as

$$s^{(m)}(\vec{x}, \theta) = \sum_{\vec{k}, \omega} s^{(m)}_{\vec{k}, \omega} \Psi_{\vec{k}, \omega}(\vec{x}, \theta). \tag{17}$$

The vector, $\mathbf{s}^{(m)}$, with components, $s^{(m)}_{\vec{k}, \omega}$, will be called the *coefficient vector* of $s^{(m)}(\vec{x}, \theta)$. In the next two sections, we demonstrate how the following integral linear transform:

$$s^{(m+1)}(\vec{x}, \theta) = \int \int \int_{\mathbf{R}^2 \times S^1} d\vec{u} d\phi \, P_0(\vec{x}, \theta \mid \vec{u}, \phi) b(\vec{u}) s^{(m)}(\vec{u}, \phi) \tag{18}$$

(i.e., the basic step in the power method) can be implemented as a discrete linear transform in a Gaussian-Fourier shiftable-twistable basis:

$$\mathbf{s}^{(m+1)} = \mathbf{PBs}^{(m)}. \tag{19}$$

## 3.3 The propagation operator P

In practice, we do not explicitly represent the matrix, $\mathbf{P}$. Instead we compute the necessary matrix-vector product using the advection-diffusion-decay operator in the Gaussian-Fourier shiftable-twistable basis, $\mathbf{A} \circ \mathbf{D}$, described in detail in Zweck and Williams[13]:

$$\mathbf{s}^{(m+1)} = \mathbf{PBs}^{(m)} \sim \lim_{n \to \infty} \mathbf{p}^{(m,n)} \tag{20}$$

where $\mathbf{p}^{(m,0)} = \mathbf{q}^{(m,0)} = \mathbf{Bs}^{(m)}$ and where:

$$\mathbf{p}^{(m,n+1)} = \mathbf{p}^{(n)} + \chi(n\Delta t) \, \mathbf{q}^{(m,n+1)} \tag{21}$$

$$\mathbf{q}^{(m,n+1)} = (\mathbf{A} \circ \mathbf{D}) \, \mathbf{q}^{(m,n)}. \tag{22}$$

In the shiftable-twistable basis, the advection operator, $\mathbf{A}$, is a discrete convolution:

$$q^{(m,n+\frac{1}{2})}_{\vec{\ell}, \eta} = \sum_{\vec{k}, \omega} \hat{a}_{\vec{\ell}-\vec{k}, \eta-\omega}(\Delta t) \, q^{(m,n)}_{\vec{k}, \omega} \tag{23}$$

with the following kernel:

$$\hat{a}_{\vec{k}, \eta}(\Delta t) = \frac{1}{2\pi} \int_0^{2\pi} a_{\vec{k}}(\Delta t [\cos \theta, \sin \theta]^{\mathrm{T}}) \, \exp(-i\eta\theta) \, d\theta \tag{24}$$

where the $a_{\vec{k}}$ are sinc functions. Let $N$ be the number of Fourier series frequencies, $\omega$, used in the shiftable-twistable basis, and let $\Delta\theta = 2\pi/N$. The diffusion-decay operator, $\mathbf{D}$, is a diagonal matrix:

$$q^{(m,n+1)}_{\vec{k}, \omega} = e^{-\Delta t/\tau} \left(\lambda e^{-i\omega \Delta\theta} + (1 - 2\lambda) + \lambda e^{i\omega \Delta\theta}\right) q^{(m,n+\frac{1}{2})}_{\vec{k}, \omega} \tag{25}$$

where $\lambda = \frac{\sigma^2}{2} \frac{\Delta t}{(\Delta\theta)^2}$.

## 3.4 The bias operator B

In the continuum, the bias operator effects a multiplication of the function, $s(\vec{x})$, by the input bias function, $b(\vec{x})$. Our aim is to identify an equivalent linear operator in the shiftable-twistable basis. Suppose that both $s$ and $b$ are represented in a Gaussian basis, $g_{\vec{k}}(\vec{x})$. Their product is:

$$b(\vec{x}) s(\vec{x}) = \sum_{\vec{k}} s_{\vec{k}} \, g_{\vec{k}}(\vec{x}) \cdot \sum_{\vec{\ell}} b_{\vec{\ell}} \, g_{\vec{\ell}}(\vec{x}) = \sum_{\vec{k}} \sum_{\vec{\ell}} s_{\vec{k}} b_{\vec{\ell}} \, g_{\vec{k}}(\vec{x}) g_{\vec{\ell}}(\vec{x}). \tag{26}$$

Now, the product of two Gaussian basis functions, $g_{\vec{k}}$ and $g_{\vec{\ell}}$, is a Gaussian of smaller variance which cannot be represented in the Gaussian basis, $g_{\vec{k}}$. Because $b(\vec{x}) s(\vec{x})$ is a linear

combination of the products of pairs of Gaussian basis functions, it cannot be represented in the Gaussian basis either. However, we observe that the convolution of $b(\vec{x})s(\vec{x})$ and a Gaussian, $h(\vec{x}) * [b(\vec{x})\,s(\vec{x})]$, where $h(\vec{x}) = \frac{1}{\Delta^2 \pi} e^{-\|\vec{x}\|^2/\Delta^2}$, can be represented in the Gaussian basis. It follows that there exists a matrix, $\mathbf{B}$, such that:

$$h(\vec{x}) * [b(\vec{x})\,s(\vec{x})] = \sum_{\vec{k}} [\mathbf{Bs}]_{\vec{k}}\, g_{\vec{k}}(\vec{x}). \tag{27}$$

The formula for the matrix, $\mathbf{B}$, is derived by first completing the square in the exponent of the product of two Gaussians to obtain:

$$g(\vec{x} - \Delta\vec{k})g(\vec{x} - \Delta\vec{\ell}) = g(\sqrt{2}(\vec{x} - \tfrac{\Delta}{2}(\vec{k} + \vec{\ell})))g(\tfrac{\Delta}{\sqrt{2}}(\vec{k} - \vec{\ell})). \tag{28}$$

This product is then convolved with $h$ to obtain a function, $f(\vec{x})$, which is a shift of the Gaussian basis function, $g(\vec{x})$. Finally we use the shiftability formula:

$$g(\vec{x} - \vec{x}_0) = \sum_{\vec{k}} a_{\vec{k}}(\vec{x}_0) g_{\vec{k}}(\vec{x}) \tag{29}$$

where $a_{\vec{k}}$ are the interpolation functions, $g_{\vec{k}}(\vec{x})$ equals $g(\vec{x} - \Delta\vec{k})$, and $\Delta = X/K$ is the shift amount, to express $f(\vec{x})$ in the Gaussian basis. The result is:

$$B_{\vec{k},\vec{\ell}} = \sum_{\vec{i}} b_{\vec{i}} \exp(-\|\vec{i} - \vec{\ell}\|^2/4) a_{\vec{k}}(\Delta(\vec{i} + \vec{\ell})/2). \tag{30}$$

## 4   Experimental results

In our experiments the Gaussian-Fourier basis consisted of $K = 192$ translates (in each spatial dimension) of a Gaussian (of period, $X = 70.0$), and $N = 92$ harmonic signals in the orientation dimension. The standard deviation of the Gaussian was set equal to the shift amount, $\Delta = X/K$. For illustration purposes, all functions were rendered at a resolution of $256 \times 256$. The diffusion parameter, $\sigma$, equaled $0.1473$, and the decay constant, $\tau$, equaled $12.5$. The time step, $\Delta t$, used to solve the Fokker-Planck equation in the basis equaled $\Delta/2$. The parameters for the cut-off function used to eliminate self-loops were $\alpha = 4$ and $\mu = 15$.

In the first experiment, the input bias function, $b(\vec{x})$, consisted of twenty randomly positioned spots and twenty spots on the boundary of an avocado. The positions of the spots are real valued, i.e., they do not lie on the grid of basis functions. See Fig. 1 (left). The stochastic completion field computed using 32 iterations of the power method is shown in Fig. 1 (right).

In the second experiment, the input bias function from the first experiment was rotated by $45°$ and translated by half the distance between the centers of adjacent basis functions, $b(R_{45°}(\vec{x} - [\tfrac{\Delta}{2}, \tfrac{\Delta}{2}]^{\mathrm{T}}))$. See Fig. 2 (left). The stochastic completion field is identical (up to rotation and translation) to the one computed in the first experiment. This demonstrates the Euclidean invariance of the computation. See Fig. 2 (right). The estimate of the largest positive real eigenvalue, $\lambda$, as a function of $m$, the power method iteration is shown in Fig. 3.

## 5   Conclusion

We described a neural network which enhances and completes salient closed contours. Even though the computation is implemented in a discrete network, its output is invariant under continuous rotations and translations of the input pattern.

## Footnotes

[1]We assume that the probability that two edges are the same depends only on the distance between them, and that $\chi(|\vec{x} - \vec{u}|) \sim \chi(t)$ for particles travelling at unit speed.

[2] We use this terminology even though the basis functions need not be linearly independent.

## References

[1] Cowan, J.D., Neurodynamics and Brain Mechanisms, *Cognition, Computation and Consciousness*, Ito, M., Miyashita, Y. and Rolls, E., (Eds.), Oxford UP, 1997.

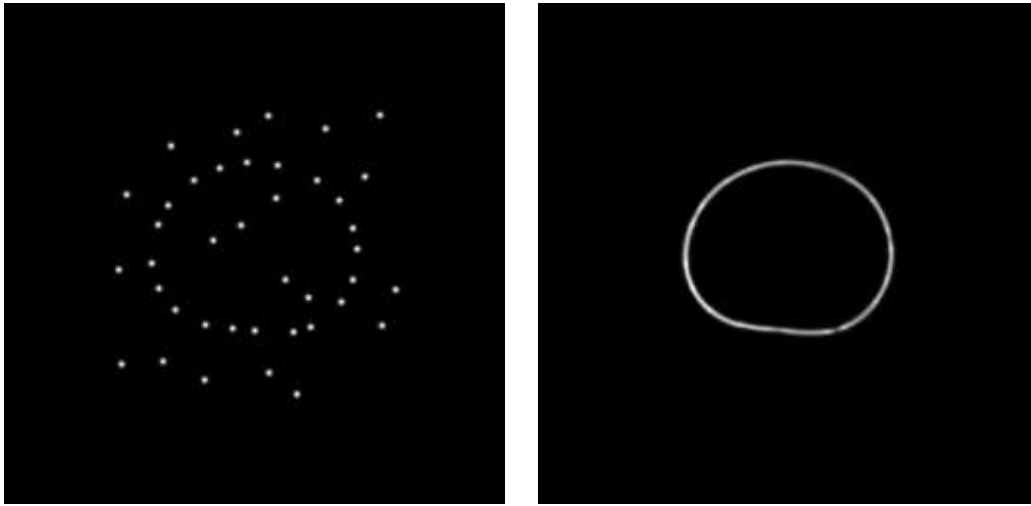

Figure 1: Left: The input bias function, $b(\vec{x})$. Twenty randomly positioned spots were added to twenty spots on the boundary of an avocado. The positions are real valued, i.e., they do not lie on the grid of basis functions. Right: The stochastic completion field, $\int_{S^1} c(\vec{u}, \phi) \, d\phi$, computed using $192 \times 192 \times 92$ basis functions.

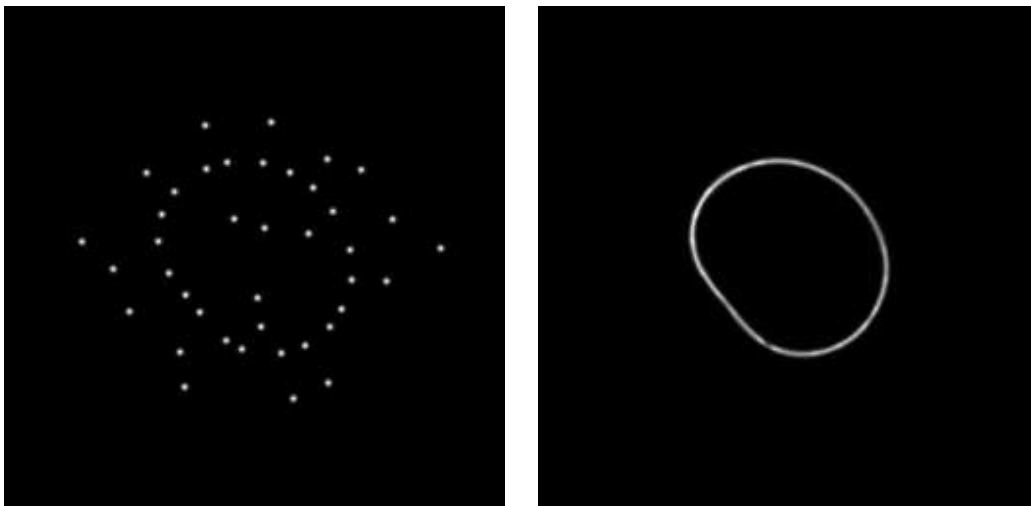

Figure 2: Left: The input bias function from Fig. 1, rotated by $45°$ and translated by half the distance between the centers of adjacent basis functions, $b(R_{45°}(\vec{x} - [\frac{\Delta}{2}, \frac{\Delta}{2}]^T))$. Right: The stochastic completion field, is identical (up to rotation and translation) to the one shown in Fig. 1. This demonstrates the Euclidean invariance of the computation.

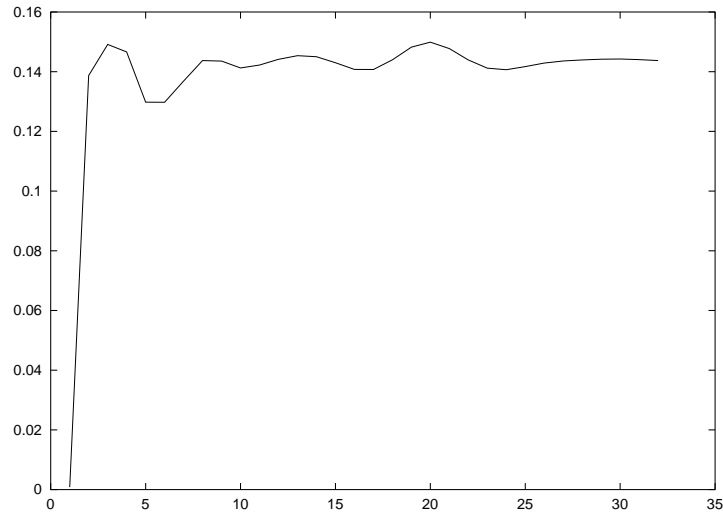

Figure 3: The estimate of the largest positive real eigenvalue, $\lambda$, as a function of $m$, the power method iteration. Both the final value and all intermediate values are identical in the rotated and non-rotated cases.

[2] Freeman, W., and Adelson, E., The Design and Use of Steerable Filters, *IEEE Transactions on Pattern Analysis and Machine Intelligence* **13** (9), pp.891-906, 1991.

[3] Mumford, D., Elastica and Computer Vision, *Algebraic Geometry and Its Applications*, Chandrajit Bajaj (ed.), Springer-Verlag, New York, 1994.

[4] Golub, G.H. and C.F. Van Loan, *Matrix Computations*, Baltimore, MD, Johns Hopkins Univ. Press, 1996.

[5] Heitger, R. and von der Heydt, R., A Computational Model of Neural Contour Processing, Figure-ground and Illusory Contours, *Proc. of 4th Intl. Conf. on Computer Vision*, Berlin, Germany, 1993.

[6] Iverson, L., Toward Discrete Geometric Models for Early Vision, Ph.D. dissertation, McGill University, 1993.

[7] Li, Z., A Neural Model of Contour Integration in Primary Visual Cortex, *Neural Computation* **10**(4), pp. 903-940, 1998.

[8] Parent, P., and Zucker, S.W., Trace Inference, Curvature Consistency and Curve Detection, *IEEE Transactions on Pattern Analysis and Machine Intelligence* **11**, pp. 823-889, 1989.

[9] Shashua, A. and Ullman, S., Structural Saliency: The Detection of Globally Salient Structures Using a Locally Connected Network, *2nd Intl. Conf. on Computer Vision*, Clearwater, FL, pp. 321-327, 1988.

[10] Simoncelli, E., Freeman, W., Adelson E. and Heeger, D., Shiftable Multiscale Transforms, *IEEE Trans. Information Theory* **38**(2), pp. 587-607, 1992.

[11] Williams, L.R., and Thornber, K.K., Orientation, Scale, and Discontinuity as Emergent Properties of Illusory Contour Shape, *Neural Computation* **13**(8), pp. 1683-1711, 2001.

[12] Yen, S. and Finkel, L., Salient Contour Extraction by Temporal Binding in a Cortically-Based Network, *Neural Information Processing Systems* **9**, Denver, CO, 1996.

[13] Zweck, J., and Williams, L., Euclidean Group Invariant Computation of Stochastic Completion Fields Using Shiftable-Twistable Functions, *Proc. European Conf. on Computer Vision (ECCV '00)*, Dublin, Ireland, 2000.
